# PRESYNAPTIC NEURAL INFORMATION PROCESSING

L. R. Carley
Department of Electrical and Computer Engineering
Carnegie Mellon University, Pittsburgh PA 15213

## ABSTRACT

The potential for presynaptic information processing within the arbor of a single axon will be discussed in this paper. Current knowledge about the activity dependence of the firing threshold, the conditions required for conduction failure, and the similarity of nodes along a single axon will be reviewed. An electronic circuit model for a site of low conduction safety in an axon will be presented. In response to single frequency stimulation the electronic circuit acts as a lowpass filter.

## I. INTRODUCTION

The axon is often modeled as a wire which imposes a fixed delay on a propagating signal. Using this model, neural information processing is performed by synaptically summing weighted contributions of the outputs from other neurons. However, substantial information processing may be performed in by the axon itself. Numerous researchers have observed periodic conduction failures at normal physiological impulse activity rates (e.g., in cat[1], in frog[2], and in man[3]). The oscillatory nature of these conduction failures is a result of the dependence of the firing threshold on past impulse conduction activity.

The simplest view of axonal (presynaptic) information processing is as a switch: the axon will either conduct an impulse or not. The state of the switch depends on how past impulse activity modulates the firing threshold, which will result in conduction failure if firing threshold is bigger than the incoming impulse strength. In this way, the connectivity of a synaptic neural network could be modulated by past impulse activity at sites of conduction failure within the network. More sophisticated presynaptic neural information processing is possible when the axon has more than one terminus, implying the existence of branch points within the axon. Section II will present a general description of potential for presynaptic information processing.

The after−effects of previous activity are able to vary the connectivity of the axonal arbor at sites of low conduction safety according to the temporal pattern of the impulse train at each site (Raymond and Lettvin, 1978; Raymond, 1979). In order to understand the information processing potential of presynaptic networks it is necessary to study the after−effects of activity on the firing threshold. Each impulse is normally followed by a brief refractory period (about 10 ms in frog sciatic nerve) of increased

threshold and a longer superexcitable period (about 1 s in frog sciatic nerve) during which the threshold is actually below its resting level. During prolonged periods of activity, there is a gradual increase in firing threshold which can persist long (> 1 hour in frog nerve) after cessation of impulse activity (Raymond and Lettvin, 1978). In section III, the methods used to measure the firing threshold and the after−effects of activity will be presented.

In addition to understanding how impulse activity modulates sites of low conduction safety, it is important to explore possible constraints on the distribution of sites of low conduction safety within the axon's arbor. Section IV presents results from a study of the distribution of the after−effects of activity along an axon.

Section V presents an electronic circuit model for a region of low conduction safety within an axonal arbor. It has been designed to have a firing threshold that depends on the past activity in a manner similar to the activity dependence measured for frog sciatic nerve.

## II. PRESYNAPTIC SIGNAL PROCESSING

Conduction failure has been observed in many different organisms, including man, at normal physiological activity rates.[1,2,3] The after−effects of activity can "modulate" conduction failures at a site of low conduction safety. One common place where the conduction safety is low is at branch points where an impedance mismatch occurs in the axon.

In order to clarify the meaning of presynaptic information processing, a simple example is in order. Parnas reported that in crayfish a single axon separately activates the medial (DEAM) and lateral (DEAL) branches of the deep abdominal extensor muscles.[4,5] At low stimulus frequencies (below 40−50 Hz) impulses travel down both branches; however, each impulse evokes much smaller contractions in DEAL than in DEAM resulting in contraction of DEAM without significant contraction of DEAL. At higher stimulus frequencies conduction in the branch leading to DEAM fails and DEAL contracts without DEAM contracting. Both DEAL and DEAM can be stimulated separately by stimulus patterns more complicated than a single frequency.

The theory of "fallible trees", which has been discussed by Lettvin, McCulloch and Pitts, Raymond, and Waxman and Grossman among others, suggests that one axon which branches many times forms an information processing element with one input and many outputs. Thus, the after−effects of previous activity are able to vary the connectivity of the axonal arbor at regions of low conduction safety according to the temporal pattern of the impulse train in each branch. The transfer function of the fallible tree is determined by the distribution of sites of low conduction safety and the distribution of superexcitability and depressibility at those sites. Thus, a single axon with 1000 terminals can potentially be in $2^{1000}$ different states as a function of the locations of sites of conduction failure within the axonal arbor. And, each site of low conduction safety is

modulated by the past impulse activity at that site.

Fallible trees have a number of interesting properties. They can be used to cause different input frequencies to excite different axonal terminals. Also, fallible trees, starting at rest, will preserve timing information in the input signal; i.e., starting from rest, all branches will respond to the first impulse.

## III. AFTER–EFFECTS OF ACTIVITY

In this section, the firing threshold will be defined and an experimental method for its measurement will be described. In addition, the after–effects of activity will be characterized and typical results of the characterization process will be given.

The following method was used to measure the firing threshold. Whole nerves were placed in the experimental setup (shown in figure 1). The whole nerve fiber was stimulated with a gross electrode. The response from a single axon was recorded using a suction microelectrode. Firing threshold was measured by applying test stimuli through the gross stimulating electrode and looking for a response in the suction microelectrode.

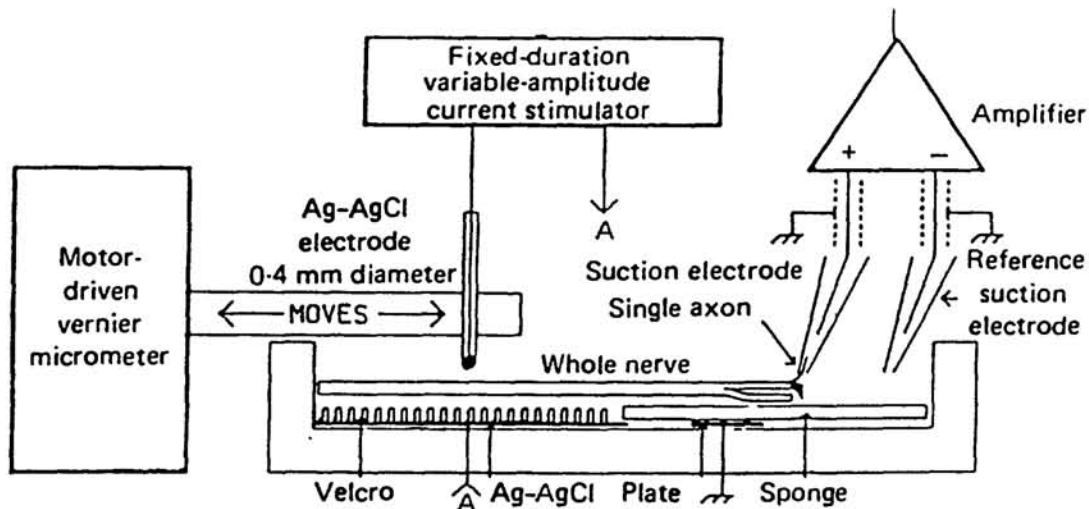

Figure 1. Drawing of the experimental recording chamber.

Threshold Hunting, a process for choosing the test stimulus strength, was used to characterize the axons.[6] It uses the following paradigm. A test stimulus which fails to elicit a conducting impulse causes a small increase the strength of subsequent test stimuli. A test stimulus which

elicits an impulse causes a small decrease in the strength of subsequent test stimuli. Conditioning Stimuli, ones large enough to guarantee firing an impulse, can be interspersed between test stimuli in order to achieve a controlled overall activity rate. Rapid variations in threshold following one or more conditioning impulses can be measured by slowly increasing the time delay between the conditioning stimuli and the test stimulus. Several phases follow each impulse. First, there is a refractory period of short duration (about 10ms in frog nerve) during which another impulse cannot be initiated. Following the refractory period the axon actually becomes more excitable than at rest for a period (ranging from 200ms to 1s in frog nerve, see figure 2). The superexcitable period is measured by applying a conditioning stimulus and then delaying by a gradually increasing time delay and applying a test stimulus (see figure 3). There is only a slight increase in the peak of the superexcitable period following multiple impulses.[7] The superexcitability of an axon was characterized by the % decrease of the threshold from its resting level at the peak of the superexcitable period.

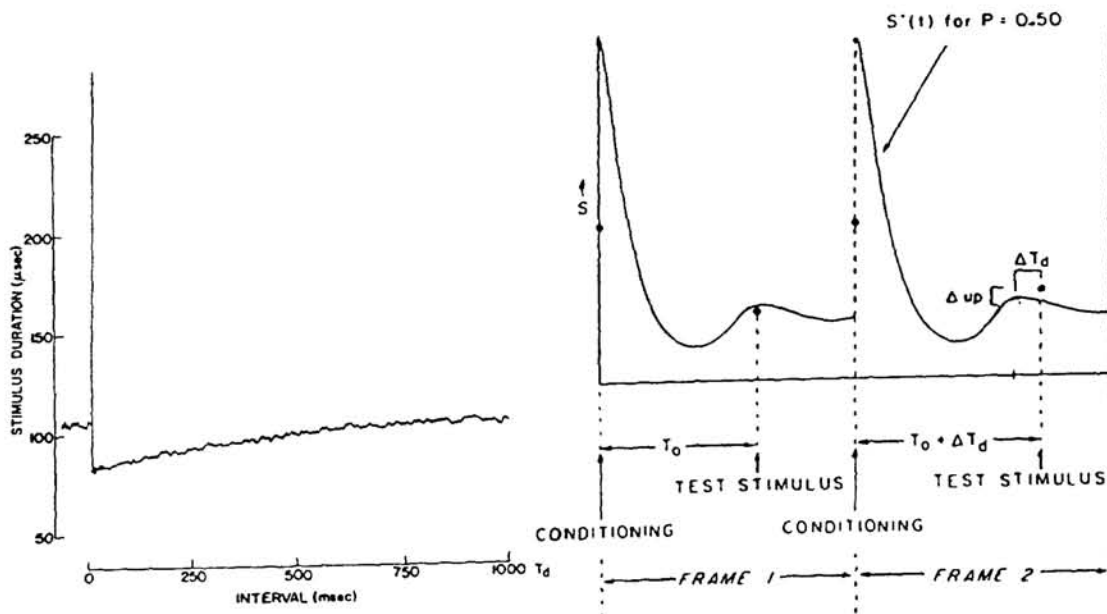

Figure 2. Typical superexcitable period in axon from frog sciatic nerve.

Figure 3. Stimulus pattern used for measuring superexcitability.

During a period of repetitive impulse conduction, the firing threshold may gradually increase. After the period of increased impulse activity ends, the threshold gradually recovers from its maximum over the course of several minutes or more with complete return of the threshold to its resting level taking as long as an hour or two (in frog nerve) depending on the extent of the preceding impulse activity. The depressibility of an axon can be characterized by the initial upward slope of the depression and the time

constant of the recovery phase (see figure 4). The pattern of conditioning and test stimuli used to generate the curve in figure 4 is shown in figure 5.

Depression may be correlated with microanatomical changes which occur in the glial cells in the nodal region during periods of increased activity.[8] During periods of repetitive stimulation the size and number of extracellular paranodal intramyelinic vacuoles increases causing changes in the paranodal geometry.

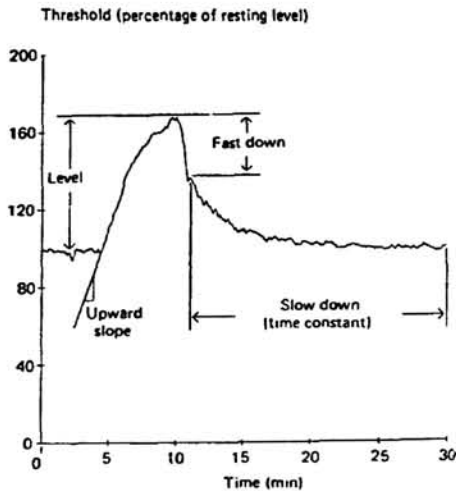

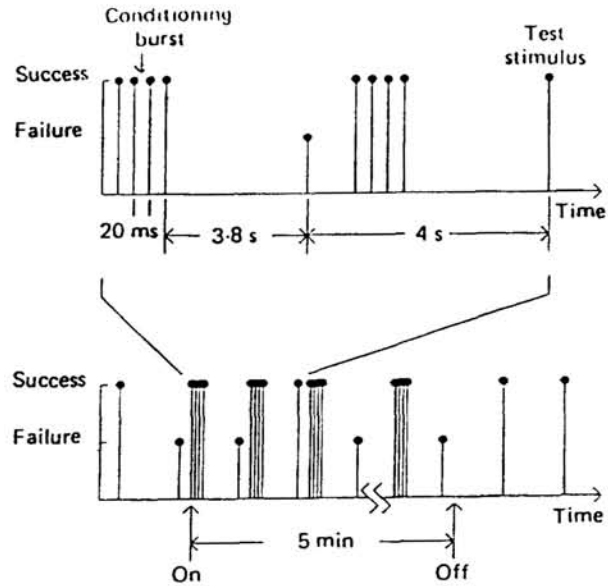

Figure 4. Typical depression in an axon from frog sciatic nerve. The average activity rate was 4 impulses/sec between the 5 min mark and the 10 min mark.

Figure 5. Stimulus pattern used for measuring depression.

## IV. CONSTRAINTS ON FALLIBLE TREES

The basic fallible tree theory places no constraints on the distribution of sites of conduction failure among the branches of a single axon. In this section one possible constraint on the distribution of sites of conduction failure will be presented. Experiments have been performed in an attempt to determine if the extremely wide variations in superexcitability and depressibility found between nodes from different axons in a single nerve[9] (particularly for depressibility) also occur between nodes from the same axon.

A study of the distribution of the after—effects of activity along an unbranching length of frog sciatic nerve found only small variations in the after—effects along a single axon.[10] Both superexcitability and depressibility were extremely consistent for nodes from along a single unbranching length of axon (see figures 6 and 7). This suggests that there may be a cell—wide regulatory system that maintains the depressibility and

superexcitability at comparable levels throughout the extent of the axon. Thus, portions of a fallible tree which have the same axon diameter would be expected to have the same superexcitability and depressibility.

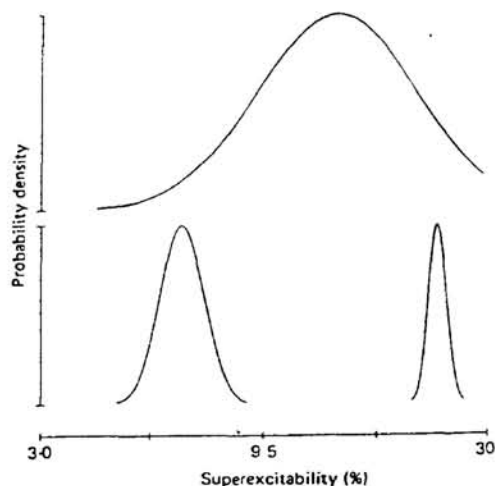
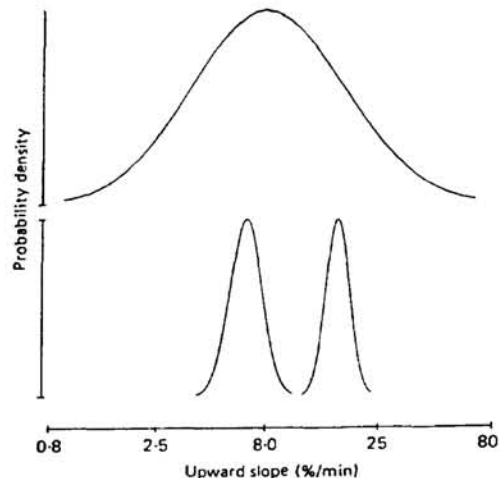

Figure 6. PDF of Superexcitability. The upper trace represents the PDF of the entire population of nodes studied and the two lower traces represent the separate populations of nodes from two different axons.

Figure 7. PDF of Depressibility. The upper trace represents the PDF of the entire population of nodes studied and the two lower traces represent the separate populations of nodes from two different axons.

This study did not examine axons which branched, therefore it cannot be concluded that superexcitability and depressibility must remain constant throughout a fallible tree. For example, it is quite likely that the cell actually regulates quantities like pump—site density, not depressibility. In that case, daughter branches of smaller diameter might be expected to show consistently higher depressibility. Further research is needed to determine how the activity dependence of the threshold scales with axon diameter along a single axon before the consistency of the after—effects along an unbranching axon can be used as a constraint on presynaptic information processing networks.

## V. ELECTRICAL AXON CIRCUIT

This section presents a simple electronic circuit which has been designed to have a firing threshold that depends on the past states of the output in a manner similar to the activity dependence measured for frog sciatic nerve. In response to constant frequency stimuli, the circuit acts as a lowpass filter whose corner frequency depends on the coefficients which determine the after–effects of activity.

Figure 8 shows the circuit diagram for a switched capacitor circuit which approximates the after–effects of activity found in the frog sciatic nerve. The circuit employs a two phase nonoverlapping clock, e for the even clock and o for the odd clock, typical of switched capacitor circuits. It incorporates a basic model for superexcitability and depressibility. $V_{TH}$ represents the resting threshold of the axon. On each clock cycle the $V_{IN}$ is compared with $V_{TH}+V_D-V_S$.

The two capacitors and three switches at the bottom of figure 8 model the change in threshold caused by superexcitability. Note that each impulse resets the comparator's minus input to $(1-\alpha_s)V_{TH}$, which decays back to $V_{TH}$ on subsequent clock cycles with a time constant inversely proportional to $\beta_S$. This is a slight deviation from the actual physiological situation in which multiple conditioning impulses will generate slightly more superexcitability than a single impulse.[7]

The two capacitors and two switches at the upper left of figure 8 model the depressibility of the axon. The current source represents a fixed increment in the firing threshold with every past impulse. The depression voltage decays back to 0 on subsequent clock cycles with a time constant inversely proportional to $\beta_D$.

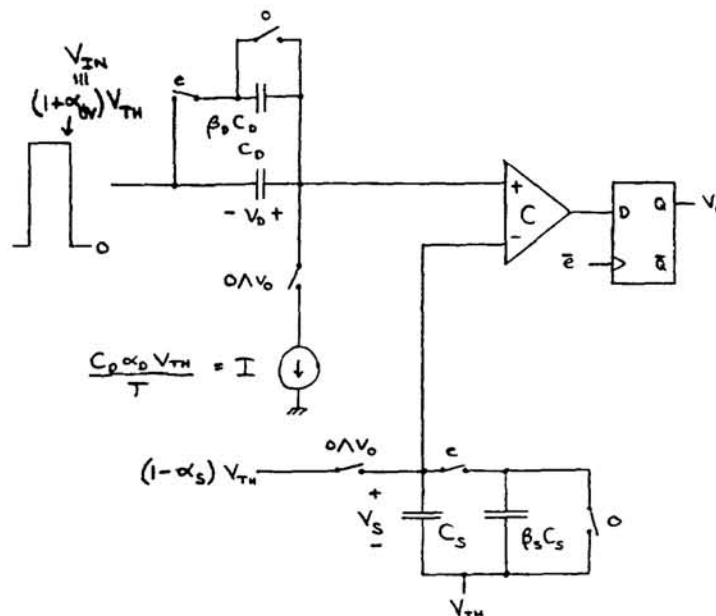

Figure 8. Circuit diagram for electrical circuit analog of nerve threshold.

The electrical circuit exhibits response patterns similar to those of neurons that are conducting intermittently (see figure 9). During bursts of conduction, the depression voltage increases linearly until the comparator

fails to fire. The electrical axon then fails to fire until the depression voltage decays back to $(1+\alpha_{OV})V_{TH}$. The connectivity between the input and output of the axon is defined to be the average fraction of impulses which are conducted. In terms of connectivity, the electrical axon model acts as a lowpass filter (see figure 10).

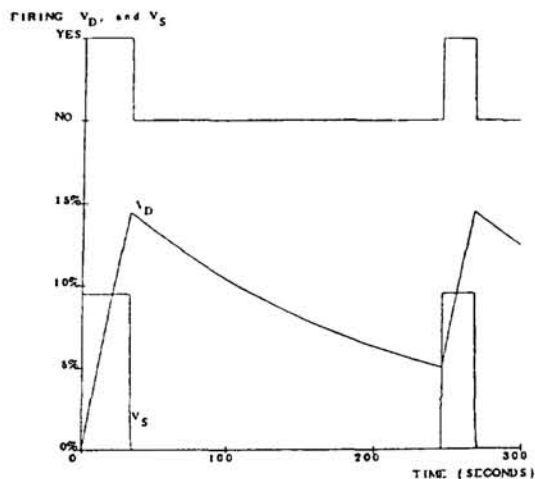

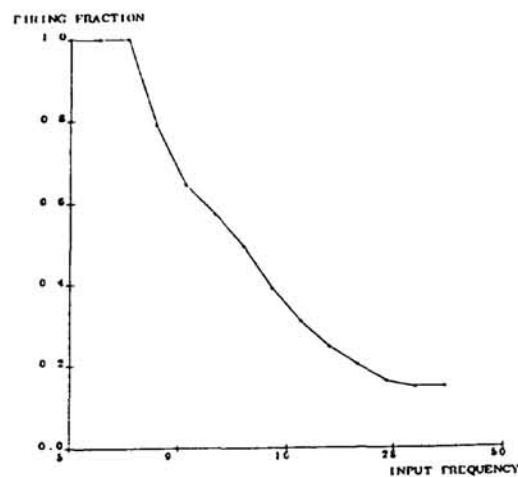

Figure 9. Typical waveforms for intermittent conduction. The upper trace indicates whether impulses are conducted or not. $V_D$ and $V_S$ are the depression voltage and the superexcitable voltage respectively.

Figure 10. Frequency response of electrical axon model. The connectivity is reflected by the fraction of impulses which are conducted out of a sequence of 100,000 stimuli where the frequency is in stimuli/second.

For a fixed stimulus frequency, the average fraction of impulses which are conducted by the electrical model can be predicted analytically. The expressions can be greatly simplified by making the assumption that $V_D$ increases and decreases in a linear fashion. Under that assumption, in terms of the variables indicated on the schematic diagram,

$$P(firing) = \frac{\alpha_{OV}(1-(1-\beta_D)^M)}{\alpha_{OV}(1-(1-\beta_D)^M)+\alpha_D}$$

where M is the number of clock cycles between input stimuli, which is inversely proportional to the input frequency. The frequency at which only half of the impulses are conducted is defined as the corner frequency of the lowpass filter. The corner frequency is

$$f(P = 0.5) = \frac{1}{M} = \frac{\log(1-\beta_D)}{\log(1-\frac{\alpha_D}{\alpha_{OV}})}$$

Using the above equations, lowpass filters with any desired cutoff frequency can be designed.

The analysis indicates that the corner frequency of the lowpass filter can be varied by changing the degree of conduction safety ($\alpha_{OV}$) without changing either depressibility or superexcitability. This suggests that the existence of a cell—wide regulatory system maintaining the depressibility and superexcitability at comparable levels throughout the extent of the axon would not prevent the construction of a bank of lowpass filters since their corner frequencies could still be varied by varying the degree of conduction safety ($\alpha_{OV}$).

## VI. CONCLUSIONS

Recent studies report that the primary effect of several common anesthetics is to abolish the activity dependence of the firing threshold without interfering with impulse conduction.[11] This suggests that presynaptic processing may play an important role in human consciousness. This paper has explored some of the basic ideas of presynaptic information processing, especially the after—effects of activity and their modulation of impulse conduction at sites of low conduction safety. A switched capacitor circuit which simulates the activity dependent conduction block that occurs in axons has been designed and simulated. Simulation results are very similar to the intermittent conduction patterns measured experimentally in frog axons. One potential information processing possibility for the arbor of a single axon, suggested by the analysis of the electronic circuit, is to act as a filterbank; every terminal could act as a lowpass filter with a different corner frequency.

## BIBLIOGRAPHY

[1] Barron D. H. and B. H. C. Matthews, Intermittent conduction in the spinal chord. *J. Physiol.* 85, p. 73—103 (1935).

[2] Fuortes M. G. F., Action of strychnine on the "intermittent conduction" of impulses along dorsal columns of the spinal chord of frogs. *J. Physiol.* 112, p.42 (1950).

[3] Culp W. and J. Ochoa, *Nerves and Muscles as Abnormal Impulse Generators.* (Oxford University Press, London, 1980).

[4] Grossman Y., I. Parnas, and M. E. Spira, Ionic mechanisms involved in differential conduction of action potentials at high frequency in a branching axon. *J. Physiol.* 295, p.307–322 (1978).

[5] Parnas I., Differential block at high frequency of branches of a single axon innervating two muscles. *J. Physiol.* 35, p. 903–914, 1972.

[6] Carley, L.R. and S.A. Raymond, Threshold Measurement: Applications to Excitable Membranes of Nerve and Muscle. *J. Neurosci. Meth.* 9, p. 309–333 (1983).

[7] Raymond S. A. and J. Y. Lettvin, After–effects of activity in peripheral axons as a clue to nervous coding. In *Physiology and Pathobiology of Axons*, S. G. Waxman (ed.), (Raven Press, New York, 1978), p. 203–225.

[8] Wurtz C. C. and M. H. Ellisman, Alternations in the ultrastructure of peripheral nodes of Ranvier associated with repetitive action potential propagation. *J. Neurosci.* 6(11), 3133–3143 (1986).

[9] Raymond S. A., Effects of nerve impulses on threshold of frog sciatic nerve fibers. *J. Physiol.* 290, 273–303 (1979).

[10] Carley, L.R. and S.A. Raymond, Comparison of the after–effects of impulse conduction on threshold at nodes of Ranvier along single frog Sciatic axons. *J. Physiol.* 386, p. 503–527 (1987).

[11] Raymond S. A. and J. G. Thalhammer, Endogenous activity–dependent mechanisms for reducing hyperexcitability of axons: Effects of anesthetics and $CO_2$. In *Inactivation of Hypersensistive Neurons*, N. Chalazonitis and M. Gola, (eds.), (Alan R. Liss Inc., New York, 1987), p. 331–343.
